# Topography and Ocular Dominance
# with Positive Correlations

**Geoffrey J. Goodhill**
University of Edinburgh
Centre for Cognitive Science
2 Buccleuch Place
Edinburgh EH8 9LW
SCOTLAND
`gjg@cns.ed.ac.uk`

## Abstract

A new computational model that addresses the formation of both topography and ocular dominance is presented. This is motivated by experimental evidence that these phenomena may be subserved by the same mechanisms. An important aspect of this model is that ocular dominance segregation can occur when input activity is both distributed, and positively correlated between the eyes. This allows investigation of the dependence of the pattern of ocular dominance stripes on the degree of correlation between the eyes: it is found that increasing correlation leads to narrower stripes. Experiments are suggested to test whether such behaviour occurs in the natural system.

## 1 INTRODUCTION

The development of topographic and interdigitated mappings in the nervous system has been much studied experimentally, especially in the visual system (e.g. [8, 15]). Here, each eye projects in a topographic manner to more central brain structures: i.e. neighbouring points in the eye map to neighbouring points in the brain. In addition, when fibres from the two eyes invade the same target struc-

ture, a competitive interaction often appears to take place such that eventually postsynaptic cells receive inputs from only one eye or the other, in a pattern of interdigitating "ocular dominance" stripes.

These phenomena have received a great deal of theoretical attention: several models have been proposed, each based on a different variant of a Hebb-type rule (e.g. [16, 17, 14, 13]). However, there are two aspects of the experimental data which previous models have not satisfactorily accounted for.

Firstly, experimental manipulations in the frog and goldfish have shown that when fibres from a second eye invade a region of brain which is normally innervated by only one eye, ocular dominance stripes can be formed (e.g. [2]). This suggests that ocular dominance may be a byproduct of the expression of the rules for topographic map formation, and does not require additional mechanisms [1]. However, previous models of topography have required additional implausible assumptions to account for ocular dominance (e.g. [16, 11], [17, 10]), while previous models of ocular dominance (e.g. [14]) have not simultaneously addressed the development of topography.

Secondly, the simulation results presented for most previous models of ocular dominance have used only localized rather than distributed patterns of input activity (e.g. [11]), or zero or negative correlations in activity between the two eyes (e.g. [3]). It is clear that, in reality, between-eye correlations have a *minimum* value of zero (which might be achieved for instance in the case of strabismus), and in general these correlations will be positive after eye-opening. In the cat for instance, the majority of ocular dominance segregation occurs anatomically three to six weeks after birth, whereas eye opening occurs at postnatal day 7-10.

Here I present a new model that accounts for (a) both topography and ocular dominance with the same mechanisms, and (b) ocular dominance segregation and receptive field refinement for input patterns which are both distributed, and positively correlated between the eyes.

## 2   OUTLINE OF THE MODEL

The model is formulated at a general enough level to be applicable to both the retinocortical and the retinotectal systems. It consists of two two dimensional sheets of input units (indexed by $r$) connected to one two-dimensional sheet of output units (indexed by $c$) by fibres with variable synaptic weights $w_{cr}$. It is assumed in the retinocortical case that the topography of the retina is essentially unchanged by the lateral geniculate nucleus (LGN) on its way to the cortex, and in addition, the effects of retinal and LGN processing are taken together. Thus for simplicity we refer to the input layers of the model as being retinae and the output layer as being the cortex. An earlier version of the model appeared in [4], and a fuller description can be found in [5].

Both retina and cortex are arranged in square arrays. All weights and unit activities are positive. Lateral interactions exist in the cortical sheet of a circular center-surround excitation/inhibition form, although these are not modeled explicitly. Initially there is total connectivity of random strengths between retinal and cortical

units, apart from a small bias that specifies an orientation for the map. At each time step, a pattern of activity is presented by setting the activities $a_r$ of retinal units. Each cortical unit $c$ calculates its total input $x_c$ according to a linear summation rule:

$$x_c = \sum_r w_{cr} a_r$$

We use assumptions similar to [9] concerning the effect of inhibitory lateral connections, to obtain the learning rule:

$$w_{cr} = w_{cr} + \alpha a_r s(c, g)$$

$\alpha$ is a small positive constant, $g$ is the cortical unit with the largest input $x_g$, and $s$ is the function that specifies how the activities of units $c$ near to $g$ decrease with distance from $g$. We assume $s$ to be a gaussian function of the Euclidean distance between units in the cortical sheet, with standard deviation $\sigma_c$.

Inputs to the model are random dot patterns with short range spatial correlation introduced by convolution with a blurring function. Locally correlated patterns of activity were generated by assigning the value 0 or 1 to each pixel in each eye with a probability of 50%, and then convolving each eye with a gaussian function of standard deviation $\sigma_r$. Between-eye correlations were produced in the following way. Once each retina has been convolved individually with a gaussian function, activity $a_j$ of each unit $j$ in each retina is replaced with $ha_j + (1 - h)a'_j$, where $a'_j$ is the activity of the corresponding unit to $j$ in the other eye, and $h$ specifies the similarity between the two eyes. Thus by varying $h$ it is possible to vary the degree of correlation between the eyes: if $h = 0$ they are uncorrelated, and if $h = 0.5$ they are perfectly correlated (i.e. the pattern of activity is identical in the two eyes).

The correlations existing in the biological system will clearly be more complicated than this. However, the simple correlational structure described above aims to capture the *key* features of the biological system: on average, cells in each retina are correlated to an extent that decreases with distance between cells, and (after eye opening) corresponding positions in the two eyes are also on the average somewhat correlated.

The sum of the weights for each postsynaptic unit is maintained at a constant fixed value. However, whereas this constraint is most usually enforced by dividing each weight by the sum of the weights for that postsynaptic unit ("divisive" normalization), it is enforced in this model by subtracting a constant amount from each weight ("subtractive" normalization), as in [13].

## 3   RESULTS

Typical results for the case of two positively correlated eyes are shown in figure 1. Gradually receptive fields refine over the course of development, and cortical units eventually lose connections from one or the other eye (figure 1(a-c)). After a large number of input patterns have been presented, cortical units are almost entirely monocular, and units dominant for the left and right eyes are laid out in a pattern of alternating stripes (figure 1(c)). In addition, maps from the two eyes are in register and topographic (figure 1(d-f)). The map of cortical receptive fields

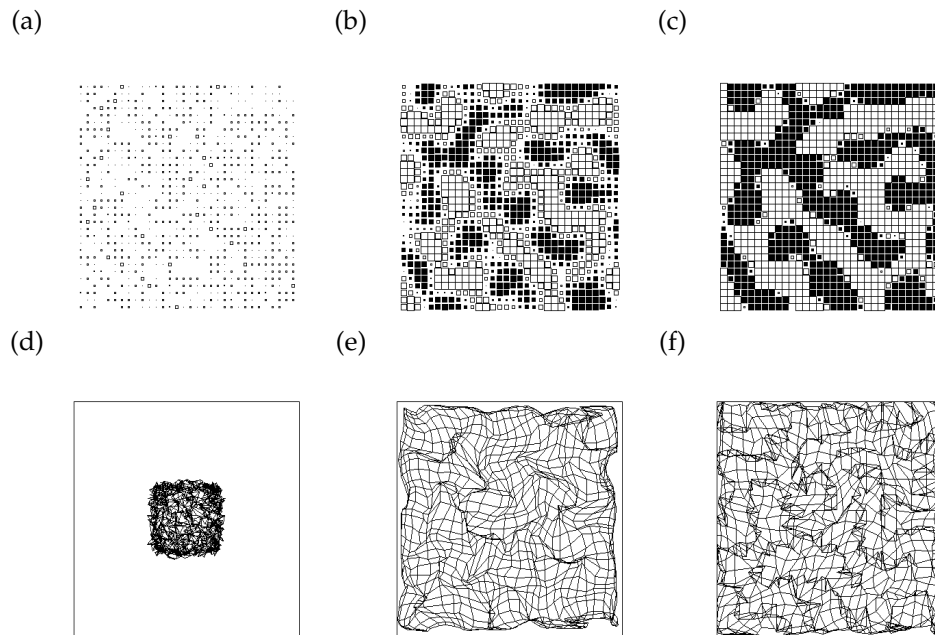

Figure 1: Typical results for two eyes. (a-c) show the ocular dominance of cortical units after 0, 50,000, and 350,000 iterations respectively. Each cortical unit is represented by a square with colour showing the eye for which it is dominant (black for right eye, white for left eye), and size showing the degree to which it is dominant. (d-f) represent cortical topography. Here the centre of mass of weights for each cortical unit is averaged over both eyes, imagining the retinae to be lying atop one another, and neighbouring units are connected by lines to form a grid. This type of picture reveals where the map is folded to take into account that the cortex must represent both eyes. It can be seen that discontinuities in terms of folds tend to follow stripe boundaries: first particular positions in one eye are represented, and then the cortex "doubles back" as its ocularity changes in order to represent corresponding positions in the other eye.

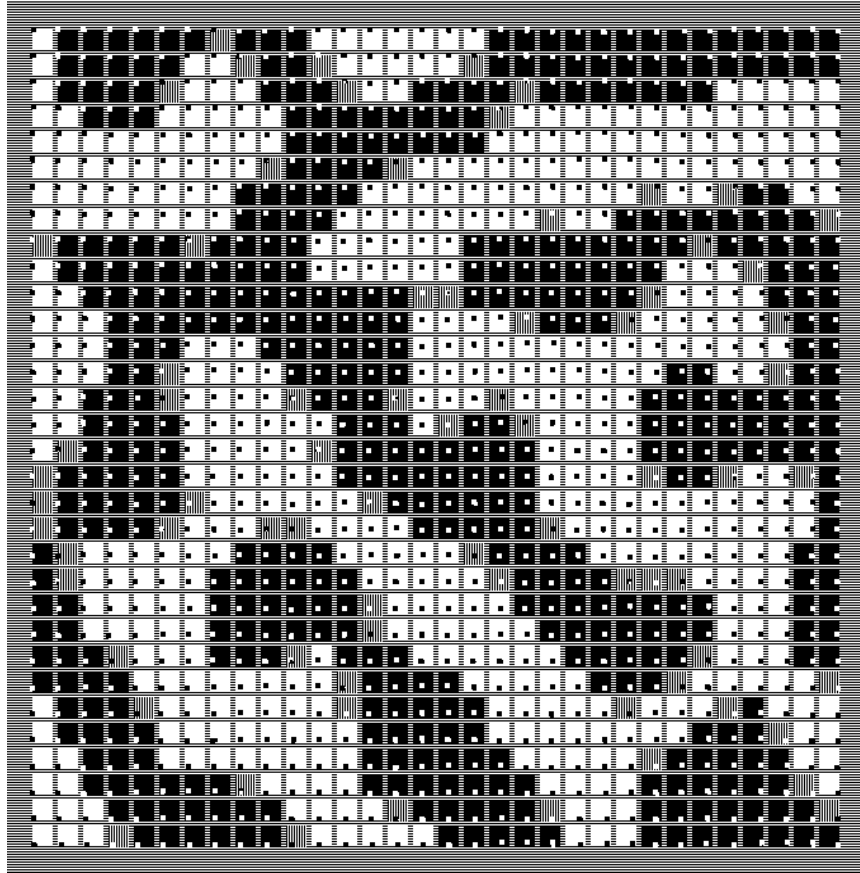

Figure 2: The receptive fields of cortical units, showing topography and eye preference. Units are coloured white if they are strongly dominant for the left eye, black if they are strongly dominant for the right eye, and grey if they are primarily binocular. "Strongly dominant" is taken to mean that at least 80% of the total weight available to a cortical unit is concentrated in one eye. Within each unit is a representation of its receptive field: there is a 16 by 16 grid within each cortical unit with each grid point representing a retinal unit, and the size of the box at each grid point encodes the strength of the connection between each retinal unit and the cortical unit. For binocular (grey) units, the larger of the two corresponding weights in the two eyes is drawn at each position, coloured white or black according to which eye that weight belongs. It can be seen that neighbouring positions in each eye tend to be represented by neighbouring cortical units, apart from discontinuities across stripe boundaries. For instance, the bottom right corner of the right retina is represented by the bottom right cortical unit, but the bottom left corner of the right retina is represented by cortical unit (3,3) (counting along and up from the bottom left corner of the cortex), since unit (1,1) represents the left retina.

(figure 2) confirms that, as described for the natural system [8], there is a smooth progression of retinal position represented across a stripe, followed by a doubling back at stripe boundaries for the cortex to "pick up where it left off" in the other eye.

An important aspect of the model is that the effect on stripe width of the strength of correlation between the two eyes can be investigated, which has not been done in previous models. Figure 3 shows a series of results for the model, from which it can be seen that stronger between-eye correlations lead to narrower stripes. It is interesting to note that a similar relationship is seen in the elastic net model of topography and ocular dominance [6], even though this is formulated on a rather different mathematical basis to the model presented here.

## 4  DISCUSSION

It has sometimes been argued that it is not necessary to also consider the development of topography in models for ocular dominance, since in the cat for instance, topography develops first, and is established before ocular dominance segregation occurs. However, non-simultaneity of development does not imply different mechanisms. As a theoretical example, in the elastic net model [6], minimisation by gradient descent of a particular objective function produces two clear stages of development: first topography formation, then ocular dominance segregation. A similar (though less marked) effect can be seen with the present model in figure 1: the rough form of the map is established before ocular dominance segregation is complete.

Interest in the contribution to map formation and eye-specific segregation of pre-visual activity in the retina has been re-awakened recently by the finding that spontaneous retinal activity takes the form of waves sweeping across the retina in a random direction [12]. Although this finding has in turn generated a wave of theoretical activity, it is important to note that the theoretical principles of how correlated activity can guide map formation have been fairly well worked out since the 1970's [16, 11]. Discovery of the precise form that these correlations take does not invalidate earlier modelling studies.

Finally, the results for the model presented in figure 3 raise the question of whether stronger between-eye correlations lead to narrower stripes in the natural system. Perhaps the simplest way to test this experimentally would be to look for changes in stripe width in the cat after artificially induced strabismus, which severely reduces the correlations between the two eyes. Although the effect of strabismus on the degree of monocularity of cortical cells has been extensively investigated (e.g. [7]), the effect on stripe width has not been examined. Such experiments would shed light on the extent to which the periodicity of ocular dominance stripes is determined by environmental as opposed to innate factors.

### Acknowledgements

This work was funded by an SERC postgraduate studentship, and an MRC/JCI postdoctoral training fellowship. I thank Harry Barrow for advice relating to this

(a)

(b)

(c)

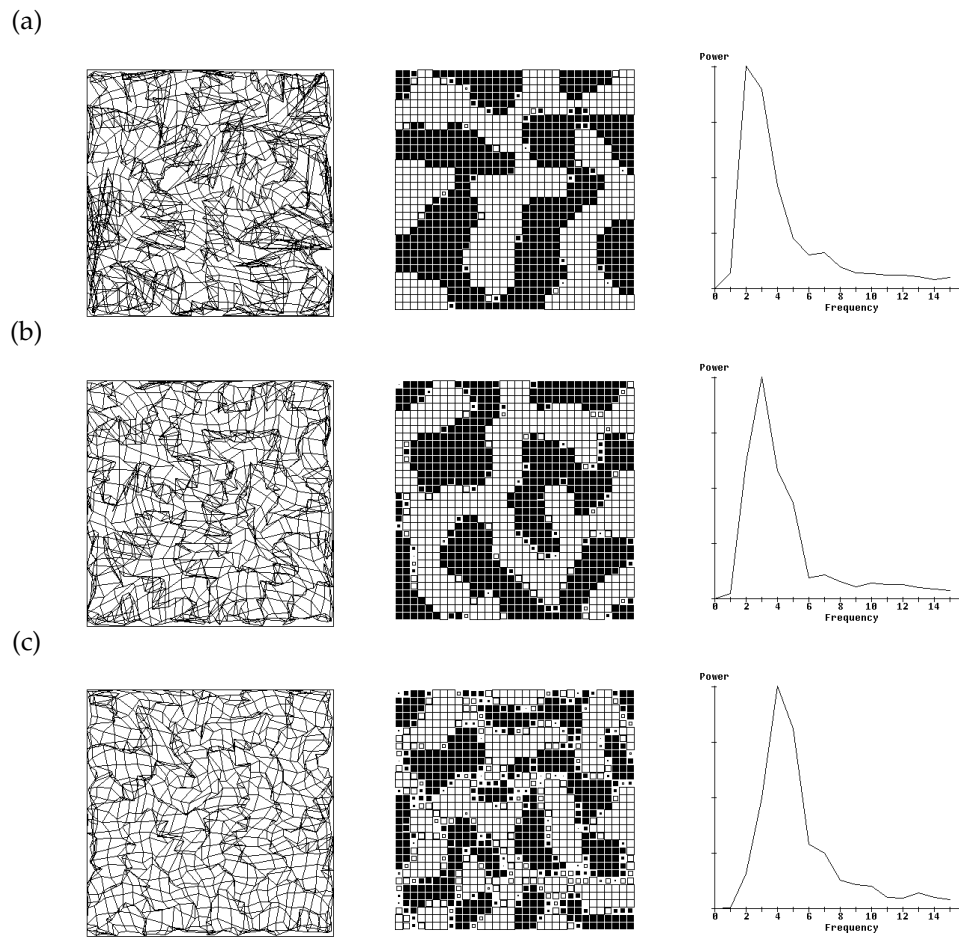

Figure 3: Effect on stripe width of the degree of correlation between the two eyes. Shown from left to right are the cortical topography averaged over both eyes, the stripe pattern, and the power spectrum of the fourier transform for each case. (a) $h = 0.0$ (b) $h = 0.1$ (c) $h = 0.2$ . Note that stripe width tends to decrease as $h$ increases, and also that the topography becomes smoother.

work, and David Willshaw and David Price for helpful comments on an earlier draft of this paper.

## References

[1] Constantine-Paton, M. (1983). Position and proximity in the development of maps and stripes. *Trends Neurosci.* **6**, 32-36.

[2] Constantine-Paton, M. & Law, M.I. (1978). Eye-specific termination bands in tecta of three-eyed frogs. *Science*, **202**, 639-641.

[3] Cowan, J.D. & Friedman, A.E. (1991). Studies of a model for the development and regeneration of eye-brain maps. In D.S. Touretzky, ed, *Advances in Neural Information Processing Systems*, **III**, 3-10.

[4] Goodhill, G.J. (1991). Topography and ocular dominance can arise from distributed patterns of activity. *International Joint Conference on Neural Networks, Seattle, July 1991*, **II**, 623-627.

[5] Goodhill, G.J. (1991). *Correlations, Competition and Optimality: Modelling the Development of Topography and Ocular Dominance.* PhD Thesis, Sussex University.

[6] Goodhill, G.J. & Willshaw, D.J. (1990). Application of the elastic net algorithm to the formation of ocular dominance stripes. *Network*, **1**, 41-59.

[7] Hubel, D.H. & Wiesel, T.N. (1965). Binocular interaction in striate cortex of kittens reared with artificial squint. *Journal of Neurophysiology*, **28**, 1041-1059.

[8] Hubel, D.H. & Wiesel, T.N. (1977). Functional architecture of the macaque monkey visual cortex. *Proc. R. Soc. Lond. B*, **198**, 1-59.

[9] Kohonen, T. (1988). Self-organization and associative memory (3rd Edition). Springer, Berlin.

[10] Malsburg, C. von der (1979). Development of ocularity domains and growth behaviour of axon terminals. *Biol. Cybern.*, **32**, 49-62.

[11] Malsburg, C. von der & Willshaw, D.J. (1976). A mechanism for producing continuous neural mappings: ocularity dominance stripes and ordered retino-tectal projections. *Exp. Brain. Res. Supplementum 1*, 463-469.

[12] Meister, M., Wong, R.O.L., Baylor, D.A. & Shatz, C.J. (1991). Synchronous bursts of action potentials in ganglion cells of the developing mammalian retina. *Science*, **252**, 939-943.

[13] Miller, K.D., Keller, J.B. & Stryker, M.P. (1989). Ocular dominance column development: Analysis and simulation. *Science*, **245**, 605-615.

[14] Swindale, N.V. (1980). A model for the formation of ocular dominance stripes. *Proc. R. Soc. Lond. B*, **208**, 243-264.

[15] Udin, S.B. & Fawcett, J.W. (1988). Formation of topographic maps. *Ann. Rev. Neurosci.*, **11**, 289-327.

[16] Willshaw, D.J. & Malsburg, C. von der (1976). How patterned neural connections can be set up by self-organization. *Proc. R. Soc. Lond. B*, **194**, 431-445.

[17] Willshaw, D.J. & Malsburg, C. von der (1979). A marker induction mechanism for the establishment of ordered neural mappings: its application to the retinotectal problem. *Phil. Trans. Roy. Soc. B*, **287**, 203-243.
